# Learning large-margin halfspaces
# with more malicious noise

**Philip M. Long**
Google
plong@google.com

**Rocco A. Servedio**
Columbia University
rocco@cs.columbia.edu

## Abstract

We describe a simple algorithm that runs in time $\text{poly}(n, 1/\gamma, 1/\varepsilon)$ and learns an unknown $n$-dimensional $\gamma$-margin halfspace to accuracy $1 - \varepsilon$ in the presence of malicious noise, when the noise rate is allowed to be as high as $\Theta(\varepsilon\gamma\sqrt{\log(1/\gamma)})$. Previous efficient algorithms could only learn to accuracy $\varepsilon$ in the presence of malicious noise of rate at most $\Theta(\varepsilon\gamma)$.

Our algorithm does not work by optimizing a convex loss function. We show that no algorithm for learning $\gamma$-margin halfspaces that minimizes a convex proxy for misclassification error can tolerate malicious noise at a rate greater than $\Theta(\varepsilon\gamma)$; this may partially explain why previous algorithms could not achieve the higher noise tolerance of our new algorithm.

## 1 Introduction

Learning an unknown halfspace from labeled examples that satisfy a margin constraint (meaning that no example may lie too close to the separating hyperplane) is one of the oldest and most intensively studied problems in machine learning, with research going back at least five decades to early seminal work on the Perceptron algorithm [5, 26, 27].

In this paper we study the problem of learning an unknown $\gamma$-margin halfspace in the model of *Probably Approximately Correct (PAC) learning with malicious noise at rate $\eta$*. More precisely, in this learning scenario the target function is an unknown origin-centered halfspace $f(\mathbf{x}) = \text{sign}(\mathbf{w} \cdot \mathbf{x})$ over the domain $\mathbf{R}^n$ (we may assume w.l.o.g. that $\mathbf{w}$ is a unit vector). There is an unknown distribution $\mathcal{D}$ over the unit ball $\mathbf{B}_n = \{\mathbf{x} \in \mathbf{R}^n : \|\mathbf{x}\|_2 \leq 1\}$ which is guaranteed to put zero probability mass on examples $\mathbf{x}$ that lie within Euclidean distance at most $\gamma$ from the separating hyperplane $\mathbf{w} \cdot \mathbf{x} = 0$; in other words, every point $\mathbf{x}$ in the support of $\mathcal{D}$ satisfies $|\mathbf{w} \cdot \mathbf{x}| \geq \gamma$. The learner has access to a *noisy example oracle* $\text{E}X_\eta(f, \mathcal{D})$ which works as follows: when invoked, with probability $1 - \eta$ the oracle draws $\mathbf{x}$ from $\mathcal{D}$ and outputs the labeled example $(\mathbf{x}, f(\mathbf{x}))$ and with probability $\eta$ the oracle outputs a "noisy" labeled example which may be an arbitrary element $(\mathbf{x}', y)$ of $\mathbf{B}_n \times \{-1, 1\}$. (It may be helpful to think of the noisy examples as being constructed by an omniscient and malevolent adversary who has full knowledge of the state of the learning algorithm and previous draws from the oracle. In particular, note that noisy examples need not satisfy the margin constraint and can lie arbitrarily close to, or on, the hyperplane $\mathbf{w} \cdot \mathbf{x} = 0$.) The goal of the learner is to output a hypothesis $h : \mathbf{R}^n \to \{-1, 1\}$ which has high accuracy with respect to $\mathcal{D}$: more precisely, with probability at least $1/2$ (over the draws from $\mathcal{D}$ used to run the learner and any internal randomness of the learner) the hypothesis $h$ must satisfy $\text{Pr}_{\mathbf{x}\sim\mathcal{D}}[h(\mathbf{x}) \neq f(\mathbf{x})] \leq \varepsilon$. (Because a success probability can be improved efficiently using standard repeat-and-test techniques [19], we follow the common practice of excluding this success probability from our analysis.) In particular, we are interested in *computationally efficient* learning algorithms which have running time $\text{poly}(n, 1/\gamma, 1/\varepsilon)$.

Introduced by Valiant in 1985 [30], the malicious noise model is a challenging one, as witnessed by the fact that learning algorithms can typically only withstand relatively low levels of malicious noise. Indeed, it is well known that for essentially all PAC learning problems it is information-theoretically possible to learn to accuracy $1 - \varepsilon$ only if the malicious noise rate $\eta$ is at most $\varepsilon/(1 + \varepsilon)$ [20], and most computationally efficient algorithms for learning even simple classes of functions can only tolerate significantly lower malicious noise rates (see e.g. [1, 2, 8, 20, 24, 28]).

Interestingly, the original Perceptron algorithm [5, 26, 27] for learning a $\gamma$-margin halfspace can be shown to have relatively high tolerance to malicious noise. Several researchers [14, 17] have established upper bounds on the number of mistakes that the Perceptron algorithm will make when run on a sequence of examples that are linearly separable with a margin except for some limited number of "noisy" data points. Servedio [28] observed that combining these upper bounds with Theorem 6.2 of Auer and Cesa-Bianchi [3] yields a straightforward "PAC version" of the online Perceptron algorithm that can learn $\gamma$-margin halfspaces to accuracy $1 - \varepsilon$ in the presence of malicious noise provided that the malicious noise rate $\eta$ is at most some value $\Theta(\varepsilon\gamma)$. Servedio [28] also describes a different PAC learning algorithm which uses a "smooth" booster together with a simple geometric real-valued weak learner and achieves essentially the same result: it also learns a $\gamma$-margin halfspace to accuracy $1 - \varepsilon$ in the presence of malicious noise at rate at most $\Theta(\varepsilon\gamma)$. Both the boosting-based algorithm of [28] and the Perceptron-based approach run in time $\text{poly}(n, 1/\gamma, 1/\varepsilon)$.

**Our results.** We give a simple new algorithm for learning $\gamma$-margin halfspaces in the presence of malicious noise. Like the earlier approaches, our algorithm runs in time $\text{poly}(n, 1/\gamma, 1/\varepsilon)$; however, it goes beyond the $\Theta(\varepsilon\gamma)$ malicious noise tolerance of previous approaches. Our first main result is:

**Theorem 1** *There is a $poly(n, 1/\gamma, 1/\varepsilon)$-time algorithm that can learn an unknown $\gamma$-margin halfspace to accuracy $1 - \varepsilon$ in the presence of malicious noise at any rate $\eta \leq c\varepsilon\gamma\sqrt{\log(1/\gamma)}$ whenever $\gamma < 1/7$, where $c > 0$ is a universal constant.*

While our $\Theta(\sqrt{\log(1/\gamma)})$ improvement is not large, it is interesting to go beyond the "natural-looking" $\Theta(\varepsilon\gamma)$ bound of Perceptron and other simple approaches. The algorithm of Theorem 1 is not based on convex optimization, and this is not a coincidence: our second main result is, roughly stated, the following.

**Informal paraphrase of Theorem 2** *Let A be any learning algorithm that chooses a hypothesis vector $\mathbf{v}$ so as to minimize a convex proxy for the binary misclassification error. Then A cannot learn $\gamma$-margin halfspaces to accuracy $1 - \varepsilon$ in the presence of malicious noise at rate $\eta \geq c\varepsilon\gamma$, where $c > 0$ is a universal constant.*

**Our approach.** The algorithm of Theorem 1 is a modification of a boosting-based approach to learning halfspaces that is due to Balcan and Blum [7] (see also [6]). [7] considers a weak learner which simply generates a random origin-centered halfspace $\text{sign}(\mathbf{v} \cdot \mathbf{x})$ by taking $\mathbf{v}$ to be a uniform random unit vector. The analysis of [7], which is for a noise-free setting, shows that such a random halfspace has probability $\Omega(\gamma)$ of having accuracy at least $1/2 + \Omega(\gamma)$ with respect to $\mathcal{D}$. Given this, any boosting algorithm can be used to get a PAC algorithm for learning $\gamma$-margin halfspaces to accuracy $1 - \varepsilon$.

Our algorithm is based on a modified weak learner which generates a collection of $k = \lceil \log(1/\gamma) \rceil$ independent random origin-centered halfspaces $h_1 = \text{sign}(\mathbf{v}_1 \cdot \mathbf{x}), \ldots, h_k = \text{sign}(\mathbf{v}_k \cdot \mathbf{x})$ and takes the majority vote $H = \mathsf{Maj}(h_1, \ldots, h_k)$. The crux of our analysis is to show that if there is no noise, then with probability at least (roughly) $\gamma^2$ the function $H$ has accuracy at least $1/2 + \Omega(\gamma\sqrt{k})$ with respect to $\mathcal{D}$ (see Section 2, in particular Lemma 1). By using this weak learner in conjunction with a "smooth" boosting algorithm as in [28], we get the overall malicious-noise-tolerant PAC learning algorithm of Theorem 1 (see Section 3).

For Theorem 2 we consider any algorithm that draws some number $m$ of samples and minimizes a convex proxy for misclassification error. If $m$ is too small then well-known sample complexity bounds imply that the algorithm cannot learn $\gamma$-margin halfspaces to high accuracy, so we may assume that $m$ is large; but together with the assumption that the noise rate is high, this means that with overwhelmingly high probability the sample will contain many noisy examples. The heart of our analysis deals with this situation; we describe a simple $\gamma$-margin data source and adversary

strategy which ensures that the convex proxy for misclassification error will achieve its minimum on a hypothesis vector that has accuracy less than $1 - \varepsilon$ with respect to the underlying noiseless distribution of examples. We also establish the same fact about algorithms that use a regularizer from a class that includes the most popular regularizers based on $p$-norms.

**Related work.** As mentioned above, Servedio [28] gave a boosting-based algorithm that learns $\gamma$-margin halfspaces with malicious noise at rates up to $\eta = \Theta(\varepsilon\gamma)$. Khardon and Wachman [21] empirically studied the noise tolerance of variants of the Perceptron algorithm. Klivans *et al.* [22] showed that an algorithm that combines PCA-like techniques with smooth boosting can tolerate relatively high levels of malicious noise provided that the distribution $\mathcal{D}$ is sufficiently "nice" (uniform over the unit sphere or isotropic log-concave). We note that $\gamma$-margin distributions are significantly less restrictive and can be very far from having the "nice" properties required by [22].

We previously [23] showed that any boosting algorithm that works by stagewise minimization of a convex "potential function" cannot tolerate random classification noise – this is a type of "benign" rather than malicious noise, which independently flips the label of each example with probability $\eta$. A natural question is whether Theorem 2 follows from [23] by having the malicious noise simply simulate random classification noise; the answer is no, essentially because the ordering of quantifiers is reversed in the two results. The construction and analysis from [23] crucially relies on the fact that in the setting of that paper, first the random misclassification noise rate $\eta$ is chosen to take some particular value in $(0, 1/2)$, and then the margin parameter $\gamma$ is selected in a way that depends on $\eta$. In contrast, in this paper the situation is reversed: in our setting first the margin parameter $\gamma$ is selected, and then given this value we study how high a malicious noise rate $\eta$ can be tolerated.

## 2   The basic weak learner for Theorem 1

Let $f(\mathbf{x}) = \mathrm{sign}(\mathbf{w} \cdot \mathbf{x})$ be an unknown halfspace and $\mathcal{D}$ be an unknown distribution over the $n$-dimensional unit ball that has a $\gamma$ margin with respect to $f$ as described in Section 1. For odd $k \geq 1$ we let $A_k$ denote the algorithm that works as follows: $A_k$ generates $k$ independent uniform random unit vectors $\mathbf{v}_1, \ldots, \mathbf{v}_k$ in $\mathbf{R}^n$ and outputs the hypothesis $H(\mathbf{x}) = \mathsf{Maj}(\mathrm{sign}(\mathbf{v}_1 \cdot \mathbf{x}), \ldots, \mathrm{sign}(\mathbf{v}_k \cdot \mathbf{x}))$. Note that $A_k$ does not use any examples (and thus malicious noise does not affect its execution). As the main result of Section 2 we show that if $k$ is not too large then algorithm $A_k$ has a non-negligible chance of outputting a reasonably good weak hypothesis:

**Lemma 1** *For odd $k \leq \frac{1}{16\gamma^2}$ the hypothesis $H$ generated by $A_k$ has probability at least $\Omega(\gamma\sqrt{k}/2^k)$ of satisfying $\mathrm{Pr}_{\mathbf{x} \sim \mathcal{D}}[H(\mathbf{x}) \neq f(\mathbf{x})] \leq \frac{1}{2} - \frac{\gamma\sqrt{k}}{100\pi}$.*

### 2.1   A useful tail bound

The following notation will be useful in analyzing algorithm $A_k$: Let $\mathrm{vote}(\gamma, k) :=$ $\mathrm{Pr}\left[\sum_{i=1}^{k} X_i < k/2\right]$ where $X_1, \ldots, X_k$ are i.i.d. Bernoulli (0/1) random variables with $\mathbf{E}[X_i] = 1/2 + \gamma$ for all $i$. Clearly $\mathrm{vote}(\gamma, k)$ is the lower tail of a Binomial distribution, but for our purposes we need an upper bound on $\mathrm{vote}(\gamma, k)$ when $k$ is very small relative to $1/\gamma^2$ and the value of $\mathrm{vote}(\gamma, k)$ is close to but – crucially – less than $1/2$. Standard Chernoff-type bounds [10] do not seem to be useful here, so we give a simple self-contained proof of the bound we need (no attempt has been made to optimize constant factors below).

**Lemma 2** *For $0 < \gamma < 1/2$ and odd $k \leq \frac{1}{16\gamma^2}$ we have $\mathrm{vote}(\gamma, k) \leq 1/2 - \frac{\gamma\sqrt{k}}{50}$.*

**Proof**: The lemma is easily verified for $k = 1, 3, 5, 7$ so we assume $k \geq 9$ below. The value $\mathrm{vote}(\gamma, k)$ equals $\sum_{i<k/2} \binom{k}{i}(1/2 - \gamma)^{k-i}(1/2 + \gamma)^i$, which is easily seen to equal $\frac{1}{2^k} \sum_{i<k/2} \binom{k}{i}(1 - 4\gamma^2)^i(1 - 2\gamma)^{k-2i}$. Since $k$ is odd $\frac{1}{2^k} \sum_{i<k/2} \binom{k}{i}$ equals $1/2$, so it remains to show that $\frac{1}{2^k} \sum_{i<k/2} \binom{k}{i}\left[1 - (1 - 4\gamma^2)^i(1 - 2\gamma)^{k-2i}\right] \geq \frac{\gamma\sqrt{k}}{50}$. Consider any integer

$i \in [0, k/2 - \sqrt{k}]$. For such an $i$ we have

$$(1 - 2\gamma)^{k-2i} \leq (1 - 2\gamma)^{2\sqrt{k}} \quad \leq \quad 1 - (2\gamma)(2\sqrt{k}) + (2\gamma)^2 \binom{2\sqrt{k}}{2} \quad (1)$$

$$\leq \quad 1 - 4\gamma\sqrt{k} + 8\gamma\sqrt{k}(\gamma\sqrt{k}) \quad (2)$$

$$\leq \quad 1 - 4\gamma\sqrt{k} + 2\gamma\sqrt{k} = 1 - 2\gamma\sqrt{k} \quad (3)$$

where (1) is obtained by truncating the alternating binomial series expansion of $(1 - 2\gamma)^{2\sqrt{k}}$ after a positive term, (2) uses the upper bound $\binom{\ell}{2} \leq \ell^2/2$, and (3) uses $\gamma\sqrt{k} \leq 1/4$ which follows from the bound $k \leq \frac{1}{16\gamma^2}$. So we have $(1 - 4\gamma^2)^i (1 - 2\gamma)^{k-2i} \leq 1 - 2\gamma\sqrt{k}$ and thus we have $1 - (1 - 4\gamma^2)^i (1 - 2\gamma)^{k-2i} \geq 2\gamma\sqrt{k}$. The sum $\sum_{i \leq k/2 - \sqrt{k}} \binom{k}{i}$ is at least $0.01 \cdot 2^k$ for all odd $k \geq 9$ [13], so we obtain the claimed bound:

$$\frac{1}{2^k} \sum_{i < k/2} \binom{k}{i} \left[1 - (1 - 4\gamma^2)^i (1 - 2\gamma)^{k-2i}\right] \geq \frac{1}{2^k} \sum_{i < k/2 - \sqrt{k}} \binom{k}{i} 2\gamma\sqrt{k} \geq \frac{\gamma\sqrt{k}}{50}. \quad \blacksquare$$

## 2.2 Proof of Lemma 1

Throughout the following discussion it will be convenient to view angles between vectors as lying the range $[-\pi, \pi)$, so acute angles are in the range $(-\pi/2, \pi/2)$.

Recall that $\mathrm{sign}(\mathbf{w} \cdot \mathbf{x})$ is the unknown target halfspace (we assume $\mathbf{w}$ is a unit vector) and $\mathbf{v}_1, \ldots, \mathbf{v}_k$ are the random unit vectors generated by algorithm $A_k$. For $j \in \{1, \ldots, k\}$ let $G_j$ denote the "good" event that the angle between $\mathbf{v}_j$ and $\mathbf{w}$ is acute, i.e. lies in the interval $(-\pi/2, \pi/2)$, and let $G$ denote the event $G_1 \wedge \cdots \wedge G_k$. Since the vectors $\mathbf{v}_i$ are selected independently we have $\Pr[G] = \prod_{j=1}^{k} \Pr[G_j] = 2^{-k}$.

The following claim shows that conditioned on $G$, any $\gamma$-margin point has a noticeably-better-than-$\frac{1}{2}$ chance of being classified correctly by $H$ (note that the probability below is over the random generation of $H$ by $A_k$):

**Claim 3** *Fix $\mathbf{x} \in \mathbf{B}_n$ to be any point such that $|\mathbf{w} \cdot \mathbf{x}| \geq \gamma$. Then we have $\Pr_H[H(\mathbf{x}) \neq f(\mathbf{x}) \mid G] \leq \mathrm{vote}(\gamma/\pi, k) \leq 1/2 - \frac{\gamma\sqrt{k}}{50\pi}$.*

**Proof:** Without loss of generality we assume that $\mathbf{x}$ is a positive example (an entirely similar analysis goes through for negative examples), so $\mathbf{w} \cdot \mathbf{x} \geq \gamma$. Let $\alpha$ denote the angle from $\mathbf{w}$ to $\mathbf{x}$ in the plane spanned by $\mathbf{w}$ and $\mathbf{x}$; again without loss of generality we may assume that $\alpha$ lies in $[0, \pi/2]$ (the case of negative angles is symmetric). In fact since $\mathbf{x}$ is a positive example with margin $\gamma$, we have that $0 \leq \alpha \leq \pi/2 - \gamma$.

Fix any $j \in \{1, \ldots, k\}$ and let us consider the random unit vector $\mathbf{v}_j$. Let $\mathbf{v}'_j$ be the projection of $\mathbf{v}_j$ onto the plane spanned by $\mathbf{x}$ and $\mathbf{w}$. The distribution of $\mathbf{v}'_j/\|\mathbf{v}'_j\|$ is uniform on the unit circle in that plane. We have that $\mathrm{sign}(\mathbf{v}_j \cdot \mathbf{x}) \neq f(\mathbf{x})$ if and only if the magnitude of the angle between $\mathbf{v}'_j$ and $\mathbf{x}$ is at least $\pi/2$. Conditioned on $G_j$, the angle from $\mathbf{v}'$ to $\mathbf{w}$ is uniformly distributed over the interval $(-\pi/2, \pi/2)$. Since the angle from $\mathbf{w}$ to $\mathbf{x}$ is $\alpha$, the angle from $\mathbf{v}'$ to $\mathbf{x}$ is the sum of the angle from $\mathbf{v}'$ to $\mathbf{w}$ and the angle from $\mathbf{w}$ to $\mathbf{x}$, and therefore it is uniformly distributed over the interval $(-\pi/2 + \alpha, \pi/2 + \alpha)$. Recalling that $\alpha \geq 0$, we have that $\mathrm{sign}(\mathbf{v}_j \cdot \mathbf{x}) \neq f(\mathbf{x})$ if and only if angle from $\mathbf{v}'$ to $\mathbf{x}$ lies in $(\pi/2, \pi/2 + \alpha)$. Since the margin condition implies $\alpha \leq \pi/2 - \gamma$ as noted above, we have $\Pr[\mathrm{sign}(\mathbf{v}_j \cdot \mathbf{x}) \neq f(\mathbf{x}) \mid G_j] \leq \frac{\pi/2 - \gamma}{\pi} = \frac{1}{2} - \frac{\gamma}{\pi}$.

Now recall that $\mathbf{v}_1, \ldots, \mathbf{v}_k$ are chosen independently at random, and $G = G_1 \wedge \cdots \wedge G_k$. Thus, after conditioning on $G$, we have that $\mathbf{v}_1, \ldots, \mathbf{v}_k$ are still independent and the events $\mathrm{sign}(\mathbf{v}_1 \cdot \mathbf{x}) \neq f(\mathbf{x}), \ldots, \mathrm{sign}(\mathbf{v}_k \cdot \mathbf{x}) \neq f(\mathbf{x})$ are independent. It follows that $\Pr_H[H(\mathbf{x}) \neq f(\mathbf{x}) \mid G] \leq \mathrm{vote}\left(\frac{\gamma}{\pi}, k\right) \leq 1/2 - \frac{\gamma\sqrt{k}}{50\pi}$, where we used Lemma 2 for the final inequality. $\blacksquare$

Now all the ingredients are in place for us to prove Lemma 1. Since Claim 3 may be applied to every $\mathbf{x}$ in the support of $\mathcal{D}$, we have $\Pr_{\mathbf{x} \sim \mathcal{D}, H}[H(\mathbf{x}) \neq f(\mathbf{x}) \mid G] \leq 1/2 - \frac{\gamma\sqrt{k}}{50\pi}$. Applying Fubini's

theorem we get that $\mathbf{E}_H[\Pr_{\mathbf{x}\sim\mathcal{D}}[H(\mathbf{x}) \neq f(\mathbf{x})] \mid G] \leq 1/2 - \frac{\gamma\sqrt{k}}{50\pi}$. Applying Markov's inequality to the nonnegative random variable $\Pr_{\mathbf{x}\sim\mathcal{D}}[H(\mathbf{x}) \neq f(\mathbf{x})]$, we get

$$\Pr_H\left[\Pr_{\mathbf{x}\sim D}[H(\mathbf{x}) \neq f(\mathbf{x})] > \frac{1 - \frac{\gamma\sqrt{k}}{50\pi}}{2} \mid G\right] \leq \frac{2(1/2 - \frac{\gamma\sqrt{k}}{50\pi})}{1 - \frac{\gamma\sqrt{k}}{50\pi}},$$

which implies

$$\Pr_H\left[\Pr_{\mathbf{x}\sim\mathcal{D}}[H(\mathbf{x}) \neq f(\mathbf{x})] \leq \frac{1 - \frac{\gamma\sqrt{k}}{50\pi}}{2} \mid G\right] \geq \Omega(\gamma\sqrt{k}).$$

Since $\Pr_H[G] = 2^{-k}$ we get

$$\Pr_H\left[\Pr_{\mathbf{x}\sim\mathcal{D}}[H(\mathbf{x}) \neq f(\mathbf{x})] \leq \frac{1 - \frac{\gamma\sqrt{k}}{50\pi}}{2}\right] \geq \Omega(\gamma\sqrt{k}/2^k),$$

and Lemma 1 is proved. ∎

## 3 Proof of Theorem 1: smooth boosting the weak learner to tolerate malicious noise

Our overall algorithm for learning $\gamma$-margin halfspaces with malicious noise, which we call Algorithm $B$, combines a weak learner derived from Section 2 with a "smooth" boosting algorithm. Recall that boosting algorithms [15, 25] work by repeatedly running a weak learner on a sequence of carefully crafted distributions over labeled examples. Given the initial distribution $P$ over labeled examples $(x, y)$, a distribution $P_i$ over labeled examples is said to be $\kappa$-*smooth* if $P_i[(x, y)] \leq \frac{1}{\kappa}P[(x, y)]$ for every $(x, y)$ in the support of $P$. Several boosting algorithms are known [9, 16, 28] that generate only $1/\varepsilon$-smooth distributions when boosting to final accuracy $1 - \varepsilon$. For concreteness we will use the MadaBoost algorithm of [9], which generates a $(1 - \varepsilon)$-accurate final hypothesis after $O(\frac{1}{\varepsilon\gamma^2})$ stages of calling the weak learner and runs in time $\text{poly}(\frac{1}{\varepsilon}, \frac{1}{\gamma})$.

At a high level our analysis here is related to previous works [28, 22] that used smooth boosting to tolerate malicious noise. The basic idea is that since a smooth booster does not increase the weight of any example by more than a $1/\varepsilon$ factor, it cannot "amplify" the malicious noise rate by more than this factor. In [28] the weak learner only achieved advantage $O(\gamma)$ so as long as the malicious noise rate was initially $O(\varepsilon\gamma)$, the "amplified" malicious noise rate of $O(\gamma)$ could not completely "overcome" the advantage and boosting could proceed successfully. Here we have a weak learner that achieves a higher advantage, so boosting can proceed successfully in the presence of more malicious noise. The rest of this section provides details.

The weak learner $W$ that $B$ uses is a slight extension of algorithm $A_k$ from Section 2 with $k = \lceil \log(1/\gamma) \rceil$. When invoked with distribution $P_t$ over labeled examples, algorithm $W$

- makes $\ell$ (specified later) calls to algorithm $A_{\lceil \log(1/\gamma) \rceil}$, generating candidate hypotheses $H_1, ..., H_\ell$; and
- evaluates $H_1, ..., H_\ell$ using $M$ (specified later) independent examples drawn from $P_t$ and outputs the $H_j$ that makes the fewest errors on these examples.

The overall algorithm $B$

- draws a multiset $S$ of $m$ examples (we will argue later that $\text{poly}(n, 1/\gamma, 1/\varepsilon)$ many examples suffice) from $EX_\eta(f, \mathcal{D})$;
- sets the initial distribution $P$ over labeled examples to be uniform over $S$; and
- uses MadaBoost to boost to accuracy $1 - \epsilon/4$ with respect to $P$, using $W$ as a weak learner.

Recall that we are assuming $\eta \leq c\varepsilon\gamma\sqrt{\log(1/\gamma)}$; we will show that under this assumption, algorithm $B$ outputs a final hypothesis $h$ that satisfies $\Pr_{\mathbf{x}\sim\mathcal{D}}[h(\mathbf{x}) = f(x)] \geq 1 - \varepsilon$ with probability at least $1/2$.

First, let $S_N \subseteq S$ denote the noisy examples in $S$. A standard Chernoff bound [10] implies that with probability at least $5/6$ we have $|S_N|/|S| \leq 2\eta$; we henceforth write $\eta'$ to denote $|S_N|/|S|$. We will show below that with high probability, every time MadaBoost calls the weak learner $W$ with a distribution $P_t$, $W$ generates a weak hypothesis (call it $h_t$) that has $\Pr_{(\mathbf{x},y)\sim P_t}[h_t(\mathbf{x}) = y] \geq 1/2 + \Theta(\gamma\sqrt{\log(1/\gamma)})$. MadaBoost's boosting guarantee then implies that the final hypothesis (call it $h$) of Algorithm B satisfies $\Pr_{(\mathbf{x},y)\sim P}[h(\mathbf{x}) = y] \geq 1 - \varepsilon/4$. Since $h$ is correct on $(1-\varepsilon/4)$ of the points in the sample $S$ and $\eta' \leq 2\eta$, $h$ must be correct on at least $1 - \varepsilon/4 - 2\eta$ of the points in $S \setminus S_N$, which is a noise-free sample of $\mathrm{poly}(n, 1/\gamma, 1/\varepsilon)$ labeled examples generated according to $\mathcal{D}$. Since $h$ belongs to a class of hypotheses with VC dimension at most $\mathrm{poly}(n, 1/\gamma, 1/\epsilon)$ (because the analysis of MadaBoost implies that $h$ is a weighted vote over $O(1/(\varepsilon\gamma^2))$ many weak hypotheses, and each weak hypothesis is a vote over $O(\log(1/\gamma))$ $n$-dimensional halfspaces), by standard sample complexity bounds [4, 31, 29], with probability $5/6$, the accuracy of $h$ with respect to $\mathcal{D}$ is at least $1 - \varepsilon/2 - 4\eta > 1 - \varepsilon$, as desired.

Thus it remains to show that with high probability each time $W$ is called on a distribution $P_t$, it indeed generates a weak hypothesis with advantage at least $\Omega(\gamma\sqrt{\log(1/\gamma)})$. Recall the following:

**Definition 1** *The* total variation distance *between distributions $P$ and $Q$ over finite domain $X$ is* $d_{TV}(P,Q) := \max_{E \subseteq X} P[E] - Q[E]$.

Suppose $R$ is the uniform distribution over the noisy points $S_N \subseteq S$, and $P'$ is the uniform distribution over the remaining points $S \setminus S_N$ (we may view $P'$ as the "clean" version of $P$). Then the distribution $P$ may be written as $P = (1 - \eta')P' + \eta'R$, and for any event $E$ we have $P[E] - P'[E] \leq \eta'R[E] \leq \eta'$, so $d_{TV}(P,P') \leq \eta'$.

Let $P_t$ denote the distribution generated by MadaBoost during boosting stage $t$. The smoothness of MadaBoost implies that $P_t[S_N] \leq 4\eta'/\epsilon$, so the noisy examples have total probability at most $4\eta'/\varepsilon$ under $P_t$. Arguing as for the original distribution, we have that the clean version $P_t'$ of $P_t$ satisfies

$$d_{TV}(P_t', P_t) \leq 4\eta'/\epsilon. \tag{4}$$

By Lemma 1, each call to algorithm $A_{\lceil \log(1/\gamma) \rceil}$ yields a hypothesis (call it $g$) that satisfies

$$\Pr_g[\mathrm{error}_{P_t'}(g) \leq 1/2 - \gamma\sqrt{\log(1/\gamma)}/(100\pi)] \geq \Omega(\gamma^2), \tag{5}$$

where for any distribution $Q$ we define $\mathrm{error}_Q(g) \stackrel{def}{=} \Pr_{(\mathbf{x},y)\sim Q}[g(\mathbf{x}) \neq y]$. Recalling that $\eta' \leq 2\eta$ and $\eta < c\varepsilon\gamma\sqrt{\log(1/\gamma)}$, for a suitably small absolute constant $c > 0$ we have that

$$4\eta'/\varepsilon < \gamma\sqrt{\log(1/\gamma)}/(400\pi). \tag{6}$$

Then (4) and (5) imply that $\Pr_g[\mathrm{error}_{P_t}(g) \leq 1/2 - 3\gamma\sqrt{\log(1/\gamma)}/(400\pi)] \geq \Omega(\gamma^2)$. This means that by taking the parameters $\ell$ and $M$ of the weak learner $W$ to be $\mathrm{poly}(1/\gamma, \log(1/\varepsilon))$, we can ensure that with overall probability at least $2/3$, at each stage $t$ of boosting the weak hypothesis $h_t$ that $W$ selects from its $\ell$ calls to $A$ in that stage will satisfy $\mathrm{error}_{P_t}(g_t) \leq 1/2 - \gamma\sqrt{\log(1/\gamma)}/(200\pi)$. This concludes the proof of Theorem 1.

## 4 Convex optimization algorithms have limited malicious noise tolerance

Given a sample $S = \{(\mathbf{x}_1, y_1), \ldots, (\mathbf{x}_m, y_m)\}$ of labeled examples, the number of examples misclassified by the hypothesis $\mathrm{sign}(\mathbf{v} \cdot \mathbf{x})$ is a nonconvex function of $\mathbf{v}$, and thus it can be difficult to find a $\mathbf{v}$ that minimizes this error (see [12, 18] for theoretical results that support this intuition in various settings). In an effort to bring the powerful tools of convex optimization to bear on various halfspace learning problems, a widely used approach is to instead minimize some convex proxy for misclassification error.

Definition 2 will define the class of such algorithms analyzed in this section. This definition allows algorithms to use regularization, but by setting the regularizer $\psi$ to be the all-0 function it also covers algorithms that do not.

**Definition 2** *A function $\phi : \mathbf{R} \to \mathbf{R}^+$ is a* convex misclassification proxy *if $\phi$ is convex, nonincreasing, differentiable, and satisfies $\phi'(0) < 0$. A function $\psi : \mathbf{R}^n \to [0, \infty)$ is a* componentwise regularizer *if $\psi(\mathbf{v}) = \sum_{i=1}^{n} \tau(v_i)$ for a convex, differentiable $\tau : \mathbf{R} \to [0, \infty)$ for which $\tau(0) = 0$. Given a sample of labeled examples $S = \{(\mathbf{x}_1, y_1), \dots, (\mathbf{x}_m, y_m)\} \in (\mathbf{R}^n \times \{-1, 1\})^m$, the $(\phi, \psi)$-loss of vector $\mathbf{v}$ on $S$ is $L_{\phi, \psi, S}(\mathbf{v}) := \psi(\mathbf{v}) + \sum_{i=1}^{m} \phi(y(\mathbf{v} \cdot \mathbf{x}_i))$. A $(\phi, \psi)$-minimizer is any learning algorithm that minimizes $L_{\phi, \psi, S}(\mathbf{v})$ whenever the minimum exists.*

Our main negative result, shows that for any sample size, algorithms that minimize a regularized convex proxy for misclassification error will succeed with exponentially small probability for a malicious noise rate that is $\Theta(\varepsilon\gamma)$, and therefore for any larger malicious noise rate.

**Theorem 2** *Fix $\phi$ to be any convex misclassification proxy and $\psi$ to be any componentwise regularizer, and let algorithm $A$ be a $(\phi, \psi)$-minimizer. Fix $\varepsilon \in (0, 1/8)$ to be any error parameter, $\gamma \in (0, 1/8]$ to be any margin parameter, and $m \geq 1$ to be any sample size. Let the malicious noise rate $\eta$ be $16\varepsilon\gamma$.*

*Then there is an $n$, a target halfspace $f(\mathbf{x}) = \mathrm{sign}(\mathbf{w} \cdot \mathbf{x})$ over $\mathbf{R}^n$, a $\gamma$-margin distribution $\mathcal{D}$ for $f$ (supported on points $\mathbf{x} \in \mathbf{B}_n$ that have $|\frac{\mathbf{w}}{\|\mathbf{w}\|} \cdot \mathbf{x}| \geq \gamma$), and a malicious adversary with the following property: If $A_\phi$ is given $m$ random examples drawn from $EX_\eta(f, \mathcal{D})$ and outputs a vector $\mathbf{v}$, then the probability (over the draws from $EX_\eta(f, \mathcal{D})$) that $\mathbf{v}$ satisfies $\mathrm{Pr}_{\mathbf{x} \sim \mathcal{D}}[\mathrm{sign}(\mathbf{v} \cdot \mathbf{x}) \neq f(\mathbf{x})] \leq \varepsilon$ is at most $e^{-c/\gamma}$, where $c > 0$ is some universal constant.*

**Proof**: The analysis has two cases based on whether or not the number of examples $m$ exceeds $m_0 := \frac{1}{32\epsilon\gamma^2}$. (We emphasize that Case 2, in which $n$ is taken to be just 2, is the case that is of primary interest, since in Case 1 the algorithm does not have enough examples to reliably learn a $\gamma$-margin halfspace even in a noiseless scenario.)

**Case 1** ($m \leq m_0$): Let $n = \lfloor 1/\gamma^2 \rfloor$ and let $\mathbf{e}^{(i)} \in \mathbf{R}^n$ denote the unit vector with a 1 in the $i$th component. Then the set of examples $E := \{\mathbf{e}^{(1)}, \dots, \mathbf{e}^{(n)}\}$ is shattered by the family $F$ which consists of all $2^n$ halfspaces whose weight vectors are in $\{-\gamma, \gamma\}^n$, and any distribution whose support is $E$ is a $\gamma$-margin distribution for any such halfspace. The proof of the well-known information-theoretic lower bound of [11][1] gives that for *any* learning algorithm that uses $m$ examples (such as $A$), there is a distribution $\mathcal{D}$ supported on $E$ and a halfspace $f \in F$ such that the output $h$ of $A$ satisfies $\mathrm{Pr}[\mathrm{Pr}_{\mathbf{x} \sim D}[h(\mathbf{x}) \neq f(\mathbf{x})] > \epsilon] \geq 1 - \exp\left(-\frac{c}{\gamma^2}\right)$, where the outer probability is over the random examples drawn by $A$. This proves the theorem in Case 1.

**Case 2** ($m > m_0$): We note that it is well known (see e.g. [31]) that $O(\frac{1}{\varepsilon\gamma^2})$ examples suffice to learn $\gamma$-margin $n$-dimensional halfspaces for any $n$ if there is no noise, so noisy examples will play an important role in the construction in this case.

We take $n = 2$. The target halfspace is $f(\mathbf{x}) = \mathrm{sign}(\sqrt{1 - \gamma^2}x_1 + \gamma x_2)$. The distribution $\mathcal{D}$ is very simple and is supported on only two points: it puts weight $2\epsilon$ on the point $\left(\frac{\gamma}{\sqrt{1-\gamma^2}}, 0\right)$ which is a positive example for $f$, and weight $1 - 2\epsilon$ on the point $(0, 1)$ which is also a positive example for $f$. When the malicious adversary is allowed to corrupt an example, with probability $1/2$ it provides the point $(1, 0)$ and mislabels it as negative, and with probability $1/2$ it provides the point $(0, 1)$ and mislabels it as negative.

Let $S = ((\mathbf{x}_1, y_1), \dots, (\mathbf{x}_m, y_m))$ be a sample of $m$ examples drawn from $EX_\eta(f, \mathcal{D})$. We define $p_{S,1} := \frac{\left|\left\{t : \mathbf{x}_t = \left(\gamma/\sqrt{1-\gamma^2}, 0\right)\right\}\right|}{|S|}$, $p_{S,2} := \frac{|\{t : \mathbf{x}_t = (0,1), y = 1\}|}{|S|}$, $\eta_{S,1} := \frac{|\{t : \mathbf{x}_t = (1,0)\}|}{|S|}$, and $\eta_{S,2} :=$

$\frac{|\{t:\mathbf{x}_t=(0,1),y=-1\}|}{|S|}$. Using standard Chernoff bounds (see e.g. [10]) and a union bound we get

$$\Pr[p_{S,1}=0 \text{ or } p_{S,2}=0 \text{ or } p_{S,1} > 3\epsilon \text{ or } \eta_{S,1} < \eta/4 \text{ or } \eta_{S,2} < \eta/4]$$

$$\leq (1-2\varepsilon(1-\eta))^m + (1-(1-2\varepsilon)(1-\eta))^m + \exp\left(-\frac{\epsilon m}{12}\right) + 2\exp\left(-\frac{\eta m}{24}\right)$$

$$\leq 2(1-\varepsilon)^m + \exp\left(-\frac{\epsilon m}{12}\right) + 2\exp\left(-\frac{\eta m}{24}\right) \qquad \text{(since } \epsilon \leq 1/4 \text{ and } \eta \leq 1/2\text{)}$$

$$\leq 2\exp\left(-\frac{1}{32\gamma^2}\right) + \exp\left(-\frac{1}{96\gamma^2}\right) + 2\exp\left(-\frac{1}{48\gamma}\right).$$

Since the theorem allows for a $e^{-c/\gamma}$ success probability for $A$, it suffices to consider the case in which $p_{S,1}$ and $p_{S,2}$ are both positive, $p_{S,1} \leq 3\epsilon$, and $\min\{\eta_{S,1}, \eta_{S,2}\} \geq \eta/4$. For $\mathbf{v} = (v_1, v_2) \in \mathbf{R}^2$ the value $L_{\phi,\psi,S}(\mathbf{v})$ is proportional to

$$L(v_1, v_2) := p_{S,1}\phi\left(\frac{\gamma v_1}{\sqrt{1-\gamma^2}}\right) + p_{S,2}\phi(v_2) + \eta_{S,1}\phi(-v_1) + \eta_{S,2}\phi(-v_2) + \frac{\psi(\mathbf{v})}{|S|}.$$

From the bounds stated above on $p_{S,1}, p_{S,2}, \eta_{S,1}$ and $\eta_{S,2}$ we may conclude that $L_{\phi,\psi,S}(\mathbf{v})$ does achieve a minimum value. This is because for any $z \in \mathbf{R}$ the set $\{\mathbf{v} : L_{\phi,\psi,S}(\mathbf{v}) \leq z\}$ is bounded, and therefore so is its closure. Since $L_{\phi,\psi,S}(\mathbf{v})$ is bounded below by zero and is continuous, this implies that it has a minimum. To see that for any $z \in \mathbf{R}$ the set $\{\mathbf{v} : L_{\phi,\psi,S}(\mathbf{v}) \leq z\}$ is bounded, observe that if either $v_1$ or $v_2$ is fixed and the other one is allowed to take on arbitrarily large magnitude values (either positive or negative), this causes $L_{\phi,\psi,S}(\mathbf{v})$ to take on arbitrarily large positive values (this is an easy consequence of the definition of $L$, the fact that $\phi$ is convex, non-negative and nonincreasing, $\phi'(0) < 0$, and the fact that $p_{S,1}, p_{S,2}, \eta_{S,1}, \eta_{S,2}$ are all positive).

Taking the derivative with respect to $v_1$ yields

$$\frac{\partial L}{\partial v_1} = p_{S,1}\frac{\gamma}{\sqrt{1-\gamma^2}}\phi'\left(\frac{\gamma v_1}{\sqrt{1-\gamma^2}}\right) - \eta_{S,1}\phi'(-v_1) - \frac{\tau'(v_1)}{|S|}. \tag{7}$$

When $v_1 = 0$, the derivative (7) is $p_{S,1}\frac{\gamma}{\sqrt{1-\gamma^2}}\phi'(0) - \eta_{S,1}\phi'(0)$ (recall that $\tau$ is minimized at 0 and thus $\tau'(0) = 0$). Recall that $\phi'(0) < 0$ by assumption. If $p_{S,1}\frac{\gamma}{\sqrt{1-\gamma^2}} < \eta_{S,1}$ then (7) is positive at 0, which means that $L(v_1, v_2)$ is an increasing function of $v_1$ at $v_1 = 0$ for all $v_2$. Since $L$ is convex, this means that for each $v_2 \in \mathbf{R}$ we have that the value $v_1^\star$ that minimizes $L(v_1^\star, v_2)$ is a negative value $v_1^\star < 0$. So, if $p_{S,1}\frac{\gamma}{\sqrt{1-\gamma^2}} < \eta_{S,1}$, the linear classifier $\mathbf{v}$ output by $A_\phi$ has $v_1 \leq 0$; hence it misclassifies the point $(\frac{\gamma}{\sqrt{1-\gamma^2}}, 0)$, and thus has error rate at least $2\epsilon$ with respect to $\mathcal{D}$.

Combining the fact that $\gamma \leq 1/8$ with the facts that $p_{S,1} \leq 3\epsilon$ and $\eta_{S,1} > \eta/4$, we get $p_{S,1}\frac{\gamma}{\sqrt{1-\gamma^2}} < 1.01 \times p_{S,1}\gamma < 4\epsilon\gamma = \eta/4 < \eta_{S,1}$ which completes the proof. ∎

## 5  Conclusion

It would be interesting to further improve on the malicious noise tolerance of efficient algorithms for PAC learning $\gamma$-margin halfspaces, or to establish computational hardness results for this problem. Another goal for future work is to develop an algorithm that matches the noise tolerance of Theorem 1 but uses a single halfspace as its hypothesis representation.

## Footnotes

[1]In particular, see the last displayed equation in the proof of Lemma 3 of [11].

## References

[1] J. Aslam and S. Decatur. Specification and simulation of statistical query algorithms for efficiency and noise tolerance. *Journal of Computer and System Sciences*, 56:191–208, 1998.

[2] P. Auer. Learning nested differences in the presence of malicious noise. *Theor. Comp. Sci.*, 185(1):159–175, 1997.

[3] P. Auer and N. Cesa-Bianchi. On-line learning with malicious noise and the closure algorithm. *Annals of Mathematics and Artificial Intelligence*, 23:83–99, 1998.

[4] E. B. Baum and D. Haussler. What size net gives valid generalization? *Neural Comput.*, 1:151–160, 1989.

[5] H. Block. The Perceptron: a model for brain functioning. *Reviews of Modern Physics*, 34:123–135, 1962.

[6] A. Blum. Random Projection, Margins, Kernels, and Feature-Selection. In *LNCS Volume 3940*, pages 52–68, 2006.

[7] A. Blum and M.-F. Balcan. A discriminative model for semi-supervised learning. *Journal of the ACM*, 57(3), 2010.

[8] S. Decatur. Statistical queries and faulty PAC oracles. In *Proc. 6th COLT*, pages 262–268, 1993.

[9] C. Domingo and O. Watanabe. MadaBoost: a modified version of AdaBoost. In *Proc. 13th COLT*, pages 180–189, 2000.

[10] D. Dubhashi and A. Panconesi. *Concentration of measure for the analysis of randomized algorithms*. Cambridge University Press, Cambridge, 2009.

[11] A. Ehrenfeucht, D. Haussler, M. Kearns, and L. Valiant. A general lower bound on the number of examples needed for learning. *Information and Computation*, 82(3):247–251, 1989.

[12] V. Feldman, P. Gopalan, S. Khot, and A. Ponnuswami. On agnostic learning of parities, monomials, and halfspaces. *SIAM J. Comput.*, 39(2):606–645, 2009.

[13] W. Feller. Generalization of a probability limit theorem of Cramér. *Trans. Am. Math. Soc.*, 54:361–372, 1943.

[14] Y. Freund and R. Schapire. Large margin classification using the Perceptron algorithm. In *Proc. 11th COLT*, pages 209–217., 1998.

[15] Y. Freund and R. Schapire. A short introduction to boosting. *J. Japan. Soc. Artif. Intel.*, 14(5):771–780, 1999.

[16] D. Gavinsky. Optimally-smooth adaptive boosting and application to agnostic learning. *JMLR*, 4:101–117, 2003.

[17] C. Gentile and N. Littlestone. The robustness of the $p$-norm algorithms. In *Proc. 12th COLT*, pages 1–11, 1999.

[18] V. Guruswami and P. Raghavendra. Hardness of learning halfspaces with noise. *SIAM J. Comput.*, 39(2):742–765, 2009.

[19] D. Haussler, M. Kearns, N. Littlestone, and M. Warmuth. Equivalence of models for polynomial learnability. *Information and Computation*, 95(2):129–161, 1991.

[20] M. Kearns and M. Li. Learning in the presence of malicious errors. *SIAM Journal on Computing*, 22(4):807–837, 1993.

[21] R. Khardon and G. Wachman. Noise tolerant variants of the perceptron algorithm. *JMLR*, 8:227–248, 2007.

[22] A. Klivans, P. Long, and R. Servedio. Learning Halfspaces with Malicious Noise. *JMLR*, 10:2715–2740, 2009.

[23] P. Long and R. Servedio. Random classification noise defeats all convex potential boosters. *Machine Learning*, 78(3):287–304, 2010.

[24] Y. Mansour and M. Parnas. Learning conjunctions with noise under product distributions. *Information Processing Letters*, 68(4):189–196, 1998.

[25] R. Meir and G. Rätsch. An introduction to boosting and leveraging. In *LNAI Advanced Lectures on Machine Learning*, pages 118–183, 2003.

[26] A. Novikoff. On convergence proofs on perceptrons. In *Proceedings of the Symposium on Mathematical Theory of Automata*, volume XII, pages 615–622, 1962.

[27] F. Rosenblatt. The Perceptron: a probabilistic model for information storage and organization in the brain. *Psychological Review*, 65:386–407, 1958.

[28] R. Servedio. Smooth boosting and learning with malicious noise. *JMLR*, 4:633–648, 2003.

[29] J. Shawe-Taylor, P. Bartlett, R. Williamson, and M. Anthony. Structural risk minimization over data-dependent hierarchies. *IEEE Transactions on Information Theory*, 44(5):1926–1940, 1998.

[30] L. Valiant. Learning disjunctions of conjunctions. In *Proc. 9th IJCAI*, pages 560–566, 1985.

[31] V. Vapnik. *Statistical Learning Theory*. Wiley-Interscience, New York, 1998.

